# Learning High-Density Regions for a Generalized Kolmogorov-Smirnov Test in High-Dimensional Data

**Assaf Glazer**
Department of Computer Science
Technion – Israel Institute of Technology
Haifa 32000, Israel
assafgr@cs.technion.ac.il

**Michael Lindenbaoum**
Department of Computer Science
Technion – Israel Institute of Technology
Haifa 32000, Israel
mic@cs.technion.ac.il

**Shaul Markovitch**
Department of Computer Science
Technion – Israel Institute of Technology
Haifa 32000, Israel
Address
shaulm@cs.technion.ac.il

## Abstract

We propose an efficient, generalized, nonparametric, statistical Kolmogorov-Smirnov test for detecting distributional change in high-dimensional data. To implement the test, we introduce a novel, hierarchical, minimum-volume sets estimator to represent the distributions to be tested. Our work is motivated by the need to detect changes in data streams, and the test is especially efficient in this context. We provide the theoretical foundations of our test and show its superiority over existing methods.

## 1 Introduction

The Kolmogorov-Smirnov (KS) test is efficient, simple, and often considered the choice method for comparing distributions. Let $\mathcal{X} = \{x_1, \ldots, x_n\}$ and $\mathcal{X}' = \{x'_1, \ldots, x'_m\}$ be two sets of feature vectors sampled i.i.d. with respect to $F$ and $F'$ distributions. The goal of the KS test is to determine whether $F \neq F'$. For one-dimensional distributions, the KS statistics are based on the maximal difference between cumulative distribution functions (CDFs) of the two distributions. However, nonparametric extensions of this test to high-dimensional data are hard to define since there are $2^{d-1}$ ways to represent a $d$-dimensional distribution by a CDF. Indeed, due to this limitation, several extensions of the KS test to more than one dimension have been proposed [17, 9] but their practical applications are mostly limited to a few dimensions.

One prominent approach of generalizing the KS test to beyond one-dimensional data is that of Polonik [18]. It is based on a generalized quantile transform to a set of high-density hierarchical regions. The transform is used to construct two sets of plots, expected and empirical, which serve as the two input CDFs for the KS test. Polonik's transform is based on a density estimation over $\mathcal{X}$. It maps the input quantile in $[0, 1]$ to a level-set of the estimated density such that the expected probability of feature vectors to lie within it is equal to its associated quantile. The expected plots are the quantiles, and the empirical plots are fractions of examples in $\mathcal{X}'$ that lie within each mapped region.

Polonik's approach can handle multivariate data, but is hard to apply in high-dimensional or small-sample-sized settings where a reliable density estimation is hard. In this paper we introduce a generalized KS test, based on Polonik's theory, to determine whether two samples are drawn from dif-

ferent distributions. However, instead of a density estimator, we use a novel hierarchical minimum-volume sets estimator to estimate the set of high-density regions directly. Because the estimation of such regions is intrinsically simpler than density estimation, our test is more accurate than density-estimation approaches. In addition, whereas Polonik's work was largely theoretical, we take a practical approach and empirically show the superiority of our test over existing nonparametric tests in realistic, high-dimensional data.

To use Polonik's generalization of the KS test, the high-density regions should be hierarchical. Using classical minimum-volume set (MV-set) estimators, however, does not, in itself, guarantee this property. We present here a novel method for approximate MV-sets estimation that guarantees the hierarchy, thus allowing the KS test to be generalized to high dimensions. Our method uses classical MV-set estimators as a basic component. We test our method with two types of estimators: *one-class SVMs (OCSVMs)* and *one-class neighbor machines (OCNMs)*.

While the statistical test introduced in this paper traces distributional changes in high dimensional data in general, it is effective in particular for change detection in data streams. Many real-world applications (e.g. *process control*) work in dynamic environments where streams of multivariate data are collected over time, during which unanticipated distributional changes in data streams might prevent the proper operation of these applications. Change-detection methods are thus required to trace such changes (e.g. [6]). We extensively evaluate our test on a collection of change-detection tasks. We also show that our proposed test can be used for the classical setting of the two-sample problem using symmetric and asymmetric variations of our test.

## 2 Learning Hierarchical High-Density Regions

Our approach for generalizing the KS test is based on estimating a hierarchical set of MV-sets in input space. In this section we introduce a method for finding such a set in high-dimensional data.

Following the notion of multivariate quantiles [8], let $\mathcal{X} = \{x_1, \ldots, x_n\}$ be a set of examples i.i.d. with respect to a probability distribution $F$ defined on a measurable space $(\mathbb{R}^d, \mathcal{S})$. Let $\lambda$ be a real-valued function defined on $\mathcal{C} \subset \mathcal{S}$. Then, the *minimum-volume set (MV-set)* with respect to $F$, $\lambda$, and $\mathcal{C}$ at level $\alpha$ is

$$C(\alpha) = \underset{C' \in \mathcal{C}}{\operatorname{argmin}} \{\lambda(C') : F(C') \geq \alpha\}. \tag{1}$$

If more than one set attains the minimum, one will be picked. Equivalently, if $F(C)$ is replaced with $F_n(C) = \frac{1}{n} \sum_1^n 1_C(x_i)$, then $C_n(\alpha)$ is one of the empirical MV-sets that attains the minimum. In the following we think of $\lambda$ as a Lebesgue measure on $\mathbb{R}^d$.

Polonik introduced a new approach that uses a hierarchical set of MV-sets to generalize the KS test beyond one dimension. Assume $F$ has a density function $f$ with respect to $\lambda$, and let $L_f(c) = \{x : f(x) \geq c\}$ be the *level set* of $f$ at level $c$. Sufficient regularity conditions on $f$ are assumed. Polonik observed that if $L_f(c) \in \mathcal{C}$, then $L_f(c)$ is an MV-set of $F$ at level $\alpha = F(L_f(c))$. He thus suggested that level-sets can be used as approximations of the MV-sets of a distribution. Hence, a density estimator was used to define a family of MV-sets $\{C(\alpha), \alpha \in [0, 1]\}$ such that a hierarchy constraint $C(\alpha) \subset C(\beta)$ is satisfied for $0 \leq \alpha < \beta \leq 1$.

We also use hierarchical MV-sets to represent distributions in our research. However, since a density estimation is hard to apply in high-dimensional data, a more practical solution is proposed. Instead of basing our method on the products of a density estimation method, we introduce a novel non-parametric method, which uses MV-set estimators (*OCSVM* and *OCNM*) as a basic component, to estimate hierarchical MV-sets without the need for a density estimation step.

### 2.1 Learning Minimum-Volume Sets with One-Class SVM Estimators

*OCSVM* is a nonparametric method for estimating a high-density region in a high-dimensional distribution [19]. Consider a function $\Phi : \mathbb{R}^d \to \mathcal{F}$ mapping the feature vectors in $\mathcal{X}$ to a hyper-sphere in an infinite Hilbert space $\mathcal{F}$. Let $\mathcal{H}$ be a hypothesis space of half-space decision functions $f_C(x) = sgn((w \cdot \Phi(x)) - \rho)$ such that $f_C(x) = +1$ if $x \in C$, and $-1$ otherwise. To separate $\mathcal{X}$

from the origin, the learner is asked to solve this quadratic program:

$$\min_{w \in \mathcal{F}, \xi \in \mathbb{R}^n, \rho \in \mathbb{R}} \frac{1}{2}||w||^2 - \rho + \frac{1}{\nu n} \sum_i \xi_i, \ \ s.t. \ \ (w \cdot \Phi(x_i)) \geq \rho - \xi_i, \ \xi_i \geq 0, \qquad (2)$$

where $\xi$ is the vector of the slack variables, and $0 < \nu < 1$ is a regularization parameter related to the proportion of outliers in the training data. All training examples $x_i$ for which $(w \cdot \Phi(x)) - \rho \leq 0$ are called *support vectors (SVs)*. Outliers are referred to as examples that strictly satisfy $(w \cdot \Phi(x)) - \rho < 0$. Since the algorithm only depends on the dot product in $\mathcal{F}$, $\Phi$ never needs to be explicitly computed, and a kernel function $k(\cdot, \cdot)$ is used instead such that $k(x_i, x_j) = (\Phi(x_i) \cdot \Phi(x_j))_{\mathcal{F}}$. The following theorem draws the connection between the $\nu$ regularization parameter and the region $C$ provided by the solution of Equation 2:

**Theorem 1** (Schölkopf et al. [19]). *Assume the solution of Equation 2 satisfies $\rho \neq 0$. The following statements hold: (1) $\nu$ is an upper bound on the fraction of outliers. (2) $\nu$ is a lower bound on the fraction of SVs. (3) Suppose $\mathcal{X}$ were generated i.i.d. from a distribution $F$ which does not contain discrete components. Suppose, moreover, that the kernel $k$ is analytic and non-constant. Then, with probability 1, asymptotically, $\nu$ is equal to both the fraction of SVs and to the fraction of outliers.*

This theorem shows that we can use *OCSVM*s to estimate high-density regions in the input space while bounding the number of examples in $\mathcal{X}$ lying outside these regions. Thus, by setting $\nu = 1 - \alpha$, we can use *OCSVM*s to estimate regions approximating $C(\alpha)$. We use this estimation method with its original quadratic optimization scheme to learn a family of MV-sets. However, a straightforward approach of training a set of *OCSVM*s, each with different $\nu \in (0, 1)$, would not necessarily satisfy the hierarchy requirement. In the following algorithm, we propose a modified construction of these regions such that both the hierarchical constraint and the density assumption (Theorem 1) will hold for each region.

Let $0 < \alpha_1 < \alpha_2, \ldots, < \alpha_q < 1$ be a sequence of quantiles. Given $\mathcal{X}$ and a kernel function $k(\cdot, \cdot)$, our *hierarchical MV-sets estimator* iteratively trains a set of $q$ *OCSVM*s, one for each quantile, and returns a set of decision functions, $\hat{f}_{C(\alpha_1)}, \ldots, \hat{f}_{C(\alpha_q)}$ that satisfy both hierarchy and density requirements. Training starts from the largest quantile ($\alpha_q$). Let $D_i$ be the training set of the *OCSVM* trained for the $\alpha_i$ quantile. Let $f_{C(\alpha_i)}, SV_{b_i}$ be the decision function and the calculated outliers (bounded SVs) of the *OCSVM* trained for the $i$-th quantile. Let $O_i = \bigcup_{j=i}^{q} SV_{b_j}$. At each iteration, $D_i$ contains examples in $\mathcal{X}$ that were not classified as outliers in previous iterations (not in $O_{i+1}$). In addition, $\nu$ is set to the required fraction of outliers over $D_i$ that will keep the total fraction of outliers over $\mathcal{X}$ equal to $1 - \alpha_i$. After each iteration, $\hat{f}_{C(\alpha_i)}$ corresponds to the intersection between the region associated with the previous decision function and the half-space associated with the current learned *OCSVM*. Thus $\hat{f}_{C(\alpha_i)}$ corresponds to the region specified by an intersection of half-spaces. The outliers in $O_i$ are points that lie strictly outside the constructed region. The pseudo-code of our estimator is given in Algorithm 1.

---

**Algorithm 1** Hierarchical MV-sets Estimator (HMVE)

---

1: **Input:** $\mathcal{X}, 0 < \alpha_1 < \alpha_2, \ldots, < \alpha_q < 1, k(\cdot, \cdot)$
2: **Output:** $\hat{f}_{C(\alpha_1)}, \ldots, \hat{f}_{C(\alpha_q)}$
3: **Initialize:** $D_q \leftarrow \mathcal{X}, O_{q+1} \leftarrow \emptyset$
4: **for** $i = q$ **to** 1 **do**
5: $\quad \nu \leftarrow \frac{(1-\alpha_i)|\mathcal{X}| - |O_{i+1}|}{|D_i|}$
6: $\quad f_{C(\alpha_i)}, SV_{b_i} \leftarrow OCSVM(D_i, \nu, k)$
7: $\quad$ **if** $i = q$ **then**
8: $\quad\quad \hat{f}_{C(\alpha_i)}(x) \leftarrow f_{C(\alpha_i(x))}$
9: $\quad$ **else**
10: $\quad\quad \hat{f}_{C(\alpha_i)}(x) \leftarrow \begin{cases} f_{C(\alpha_i(x))} & : \quad \hat{f}_{C(\alpha_{i+1})}(x) \\ -1 & : \quad \text{otherwise} \end{cases}$
11: $\quad O_i \leftarrow O_{i+1} \cup SV_{b_i}, D_{i-1} \leftarrow D_i \setminus SV_{b_i}$
12: **return** $\hat{f}_{C(\alpha_1)}, \ldots, \hat{f}_{C(\alpha_q)}$

---

The following theorem shows that the regions specified by the decision functions $\hat{f}_{C(\alpha_1)}, \ldots, \hat{f}_{C(\alpha_q)}$ are: (a) approximations for the MV-sets in the same sense suggested by Schölkopf et al., and (b) hierarchically nested. In the following, $\hat{C}(\alpha_i)$ is denoted as the estimates of $C(\alpha_i)$ with respect to $\hat{f}_{C(\alpha_i)}$.

**Theorem 2.** *Let $\hat{f}_{C(\alpha_1)}, \ldots, \hat{f}e_{C(\alpha_q)}$ be the decision functions returned by Algorithm 1 with parameters $\{\alpha_1, \ldots, \alpha_q\}, \mathcal{X}, k(\cdot, \cdot)$. Assume $\mathcal{X}$ is separable. Let $\hat{C}(\alpha_i)$ be the region in the input space*

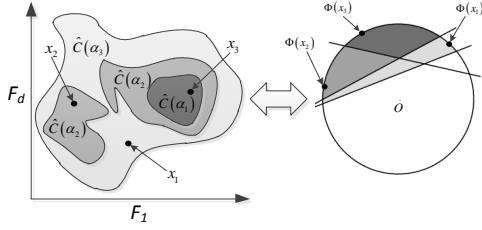

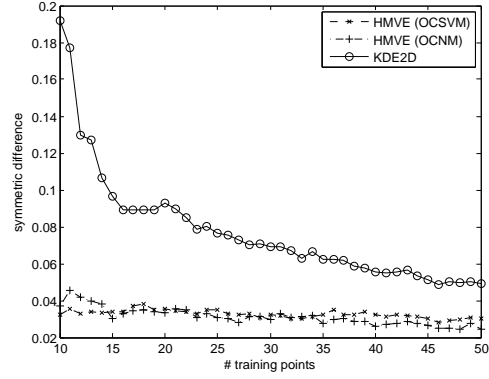

Figure 1: Left: Estimated MV-sets $\hat{C}(\alpha_i)$ in the original input space, $q = 3$. Right: the projected $\hat{C}(\alpha_i)$ in $\mathcal{F}$.

Figure 2: Averaged symmetric differences against the number of training points for the *OCSVM / OCNM* versions of our estimator, and the KDE2d density estimator

associated with $\hat{f}_{C(\alpha_i)}$, and $SV_{ub_i}$ be the set of (unbounded) SVs lying on the separating hyperplane in the region associated with $f_{C(\alpha_i(x))}$. Then, the following statements hold:(1) $\hat{C}(\alpha_i) \subseteq \hat{C}(\alpha_j)$ for $\alpha_i < \alpha_j$. (2) $\frac{|O_i|}{|\mathcal{X}|} \leq 1 - \alpha_i \leq \frac{|SV_{ub_i}|+|O_i|}{|\mathcal{X}|}$. (3) Suppose $\mathcal{X}$ were i.i.d. drawn from a distribution $F$ which does not contain discrete components, and $k$ is analytic and non-constant. Then, $1 - \alpha_i$ is asymptotically equal to $\frac{|O_i|}{|\mathcal{X}|}$.

*Proof.* Statement (1) holds by definition of $\hat{f}_{C(\alpha_i)}$. Statements (2)-(3) are proved by induction on the number of iterations. In the first iteration $\hat{f}_{C(\alpha_q)}$ equals $f_{C(\alpha_q)}$. Thus, since $O_q = SV_{b_q}$ and $\nu = 1 - \alpha_q$, statements (2)-(3) follow directly from Theorem 1 [1]. Then, by the induction hypothesis, statements (2)-(3) hold for the first $n-1$ iterations over the $\alpha_q, \ldots, \alpha_{q-n+1}$ quantiles. We now prove that statements (2)-(3) hold for $\hat{f}_{C(\alpha_{q-n})}$ in the next iteration. Since $\hat{f}_{C(\alpha_{q-n+1})}(x) = -1$ implies $\hat{f}_{C(\alpha_{q-n})}(x) = -1$, $O_{q-n+1}$ are outliers with respect to $\hat{f}_{C(\alpha_{q-n})}$. In addition, $\nu = \frac{(1-\alpha_{q-n})|\mathcal{X}|-|O_{q-n+1}|}{|D_i|}$. Hence, following Theorem 1, the total proportion of outliers with respect to $\mathcal{X}$ is $|O_{q-n}| = |SV_{b_{q-n}}| + |O_{q-n+1}| \leq \nu|D_i| + |O_{q-n+1}| = (1 - \alpha_{q-n})|\mathcal{X}|$, and $|SV_{ub_{q-n}}| + |O_{q-n+1}| \geq (1 - \alpha_{q-n})|\mathcal{X}|$. Hence, $\frac{|O_{q-n}|}{|\mathcal{X}|} \leq 1 - \alpha_{q-n} \leq \frac{|SV_{ub_{q-n}}|+|O_{q-n}|}{|\mathcal{X}|}$. In the same manner, under the conditions of statement (3), $|O_{q-n}|$ is asymptotically equal to $(1 - \alpha_{q-n})|\mathcal{X}|$, and hence, asymptotically, $1 - \alpha_{q-n} = \frac{|O_{q-n}|}{|\mathcal{X}|}$. $\qquad\square$

Figure 1 illustrates the estimated MV-sets $\hat{C}(\alpha_i)$ in both the original and the projected spaces. On the left, all $\hat{C}(\alpha_i)$ regions in the original input space are colored with decreased gray levels. Note that $\hat{C}(\alpha_i)$ is a subset of $\hat{C}(\alpha_j)$ if $i < j$. On the right, the projected regions of all $\hat{C}(\alpha_i)$s in $\mathcal{F}$ are marked with the same colors. Examples $x_i$ in the input space and their mapped vectors $\phi(x_i)$ in $\mathcal{F}$ are contained in the same relative regions in both spaces. It can be seen that the projections of $\hat{C}(\alpha_i)$ in $\mathcal{F}$ are the intersecting half-spaces learned by Algorithm 1.

## 2.2 Learning Minimum-Volume Sets with One-Class Neighbor Machine Estimators

*OCNM* [15] is as an alternative method to the *OCSVM* estimator for finding regions close to $C(\alpha)$. Unlike *OCSVM*, the *OCNM* solution is proven to be asymptotically close to the MV-set specified [2]. Degenerated structures in data that may damage the generalization of SVMs could be another reason for choosing *OCNM* [24]. In practice, for finite sample size, it is not clear which estimator is more accurate.

*OCNM* uses either a sparsity or a concentration neighborhood measure. $M(\mathbb{R}^d, \mathcal{X}) \to \mathbb{R}$ is a *sparsity measure* if $f(x) > f(y)$ implies $lim_{|\mathcal{X}| \to \infty} P(M(x, \mathcal{X}) < M(y, \mathcal{X})) = 1$. An example for a valid sparsity measure is the distance of $x$ to its $k$th-nearest neighbor in $\mathcal{X}$. When a sparsity measure is used, the *OCNM* estimator solves the following linear problem

$$\max_{\xi \in \mathbb{R}^n, \rho \in \mathbb{R}} \nu n \rho - \sum_i^n \xi_i, \ \ s.t. \ M(x_i, \mathcal{X}) \geq \rho - \xi_i, \ \xi_i \geq 0, \tag{3}$$

such that the resulting decision function $f_C(x) = sgn(\rho - M(x, \mathcal{X}))$ satisfies bounds and convergence properties similar to those mentioned in Theorem 1 ($\nu$-property).

*OCNM* can replace *OCSVM* in our hierarchical MV-sets estimator. In contrast to *OCSVM*s, when *OCNM*s are iteratively trained on $\mathcal{X}$ using a growing sequence of $\nu$ values, outliers need not be moved from previous iterations to ensure that the $\nu$-property will hold for each decision function. Hence, a simpler version of Algorithm 1 can be used, where $\mathcal{X}$ is used for training all *OCNM*s and $\nu = 1 - \alpha_i$ for each step [3]. Since Theorem 2 relies on the $\nu$-property of the estimator, it can be shown that similar statements to those of Theorem 2 also hold when *OCNM* is used.

As previously discussed, since the estimation of MV-sets is simpler than density estimation, our test can achieve higher accuracy than approaches based on density estimation. To illustrate this hypothesis empirically, we conducted the following preliminary experiment. We sampled 10 to 50 i.i.d. points with respect to a two-dimensional, mixture of Gaussians, distribution $p = \frac{1}{2}\mathcal{N}(\mu = (0.5, 0.5), \Sigma = 0.1I) + \frac{1}{2}\mathcal{N}(\mu = (-0.5, -0.5), \Sigma = 0.5I)$. We use the *OCNM* and *OCSVM* versions of our estimator to approximate hierarchical MV-sets for $q_\alpha = 9$ quantiles: $\alpha = 0.1, 0, 2, \dots, 0.9$ (detailed setup parameters are discussed in Section 4). MV-sets estimated with a KDE2d kernel-density estimation [2] were used for comparison. For each sample size, we measured the error of each method according to the mean weighted symmetric difference between the true MV-sets and their estimates, $\frac{1}{q_\alpha} \sum_\alpha \int_{x \in C(\alpha) \Delta \hat{C}(\alpha)} p(x) dx$. Results, averaged over 50 simulations, are shows in Figure 2. The advantages of our approach can easily be seen: both versions of our estimator preform notably better, especially for small sample sizes.

## 3 Generalized Kolmogorov-Smirnov Test

We now introduce a nonparametric, *generalized Kolmogorov-Smirnov (GKS) statistical test* for determining whether $F \neq F'$ in high-dimensional data. Assume $F, F'$ are one-dimensional continuous distributions and $F_n, F'_m$ are empirical distributions estimated from $n$ and $m$ examples i.i.d. drawn from $F, F'$. Then, the two-sample Kolmogorov-Smirnov (KS) statistic is

$$\text{KS}_{n,m} = \sup_{x \in \mathcal{R}} |F_n(x) - F'_m(x)| \tag{4}$$

and $\sqrt{\frac{nm}{n+m}} \text{KS}_{n,m}$ is asymptotically distributed, under the null hypothesis, as the distribution of $\sup_{x \in \mathcal{R}} |B(F(x))|$ for a standard Brownian bridge $B$ when $F = F'$. Under the null hypothesis, assume $F = F'$ and let $F^{-1}$ be a quantile transform of $F$, i.e., the inverse of $F$. Then we can replace the supremum over $x \in \mathcal{R}$ with the supremum over $\alpha \in [0, 1]$ as follows:

$$\text{KS}_{n,m} = \sup_{\alpha \in [0,1]} \left| F_n(F^{-1}(\alpha)) - F'_m(F^{-1}(\alpha)) \right|. \tag{5}$$

Note that in the one-dimensional setting, $F^{-1}(\alpha)$ is the point $x$ *s.t.* $F(X \leq x) \leq \alpha$ where $X$ is a random variable drawn from $F$. Equivalently, $F^{-1}(\alpha)$ can be identified with the interval $[-\infty, x]$. In a high-dimensional space these intervals can be replaced by hierarchical MV-sets $C(\alpha)$ [18], and hence, Equation 5 can be calculated regardless of the input space dimensionality. We suggest replacing $\text{KS}_{n,m}$ with

$$\text{T}_{n,m} = \sup_{\alpha \in [0,1]} |F_n(C(\alpha)) - F'_m(C(\alpha))|. \tag{6}$$

For estimating $C(\alpha)$ we use our nonparametric method from Section 2. $\hat{C}(\alpha)$ is learned with $\mathcal{X}$ and marked as $\hat{C}_\mathcal{X}(\alpha)$. In practice, when $|\mathcal{X}|$ is finite, the expected proportion of examples that lie

within $\hat{C}_\mathcal{X}(\alpha_i)$ is not guaranteed to be exactly $\alpha_i$. Therefore, after learning the decision functions, we estimate $F_n(\hat{C}_\mathcal{X}(\alpha_i))$ by a *k-folds cross-validation* procedure. Our final test statistic is

$$\hat{T}_{n,m} = \sup_{1 \le i \le q} \left| \hat{F}_n(\hat{C}_\mathcal{X}(\alpha_i)) - F_m(\hat{C}_\mathcal{X}(\alpha_i)) \right|, \tag{7}$$

where $\hat{F}_n(\hat{C}_\mathcal{X}(\alpha_i))$ is the estimate of $F_n(\hat{C}_\mathcal{X}(\alpha_i))$. The two-sample KS statistical test is used over $\hat{T}_{n,m}$ to calculate the resulting *p*-value.

The test defined above works only in one direction by predicting whether distributions of the samples share the same "concentrations" as regions estimated according to $\mathcal{X}$, and not according to $\mathcal{X}'$. We may symmetrize it by running the non-symmetric test twice, once in each direction, and return twice their minimum *p*-value (*Bonferroni correction*). Note that by doing so in the context of a change detection task, we pay in runtime required for learning MV-sets for each $\mathcal{X}'$.

## 4 Empirical Evaluation

We first evaluated our test on concept-drift detection problems in data-stream classification tasks. Concept drifts are associated with distributional changes in data streams that occur due to *hidden context* [22] — changes of which the classifier is unaware. We used the 27 UCI datasets used in [6], and 6 additional high-dimensionality UCI datasets: *arrhythmia*, *madelon*, *semeion*, *internet advertisement*, *hill-valley*, and *musk*. The average number of features for all datasets is 123 [4].

Following the experimental setup used by [11, 6], we generated, for each dataset, a sequence $\langle x_1, \ldots, x_{n+m} \rangle$, where the first $n$ examples are associated with the most frequent label, and the following $m$ examples with the second most frequent. Within each label the examples were shuffled randomly. The first 100 examples $\langle x_1, \ldots, x_{100} \rangle$, associated, in all datasets, with the most common label, were used as the baseline dataset $\mathcal{X}$. A sliding window of 50 consecutive examples over the following sequence of examples was iteratively used to define the most recent data $\mathcal{X}'$ at hand. Statistical tests were evaluated with $\mathcal{X}$ and all possible $\mathcal{X}'$ windows. In total, for each dataset, the set $\{\langle \mathcal{X}, \mathcal{X}'_i \rangle | \mathcal{X}'_i = \{x_i, \ldots, x_{i+49}\}, 101 \le i \le n + m - 49\}$ of pairs were used for evaluation. The following figure illustrates this setup:

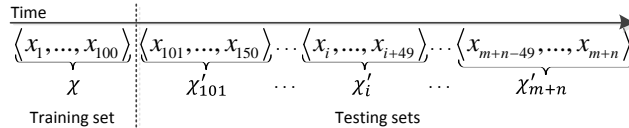

The pairs $\langle \mathcal{X}, \mathcal{X}'_i \rangle, i \le n - 49$, where all examples in $\mathcal{X}'_i$ have the same labels as in $\mathcal{X}$, are considered "unchanged." The remaining pairs are considered "changed." Performance is evaluated using precision-recall values with respect to the change detection task.

We compare our one-directional ($GKS_{1d}$) and two-directional ($GKS_{2d}$) tests to the following 5 reference tests: *kdq-tree test (KDQ)* [4], *Metavariable Wald-Wolfowitz test (WW)* [10], *Kernel change detection (KCD)* [5], *Maximum mean discrepancy test (MMD)* [12], and *PAC-Bayesian margin test (PBM)* [6]. See section 5 for details. All tests, except of *MMD*, were implemented and parameters were set with accordance to their suggested setting in their associate papers. The implementation of *MMD* test provided by the authors [5] was used with default parameters (RBF kernels with automatic kernel width detection) and Rademacher bounds. Similar results were also measured for asymptotic bounds. Note that we cannot compare our test to Polonik's test since density estimations and level-sets extractions are not practically feasible on high-dimensional data.

The *LibSVM* package [3] with a Gaussian kernel ($\gamma = \frac{2}{\#features}$) was used for the *OCSVM*s. A distance from a point to its $k$th-nearest neighbor was used as a sparsity measure for the *OCNM*s. $k$ is set to 10% of the sample size [6]. $\alpha = 0.1, 0.2, \ldots, 0.9$ were used for all experiments.

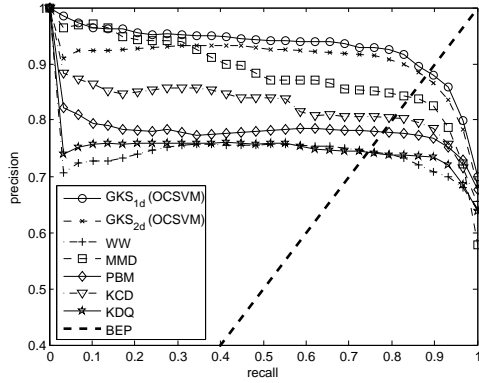
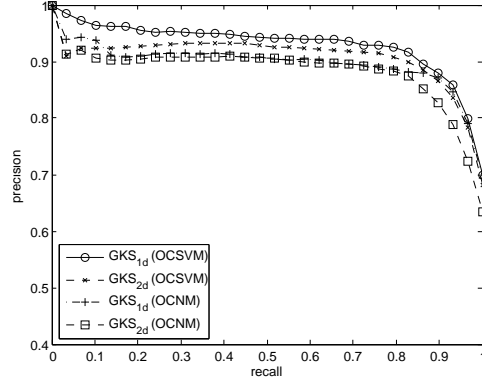

Figure 3: Precision-recall curves averaged over all 33 experiments for $GKS_{1d}$ (OCSVMs), $GKS_{2d}$ (OCSVMs), and the 5 reference tests.

Figure 4: Precision-recall curves averaged over all 33 experiments for $GKS_{1d}$ (OCSVMs), $GKS_{2d}$ (OCSVMs), $GKS_{1d}$ (OSNMs), and $GKS_{2d}$ (OSNMs).

## 4.1 Results

For better visualization, results are shown in two separate figures: Figure 3 shows the precision-recall plots averaged over the 33 experiments for the *OCSVM* version of our tests, and the 5 reference tests. Figure 4 shows the precision-recall plots averaged over the 33 experiments for the *OCSVM* and *OCNM* versions of our tests. In both versions, $GKS_{1d}$ and $GKS_{2d}$ provide the best precision-recall compromise. For example, for the *OCSVM* version, at a recall of 0.86, $GKS_{1d}$ accurately detects distributional changes with 0.90 precision and $GKS_{2d}$ with 0.88 precision, while the second best competitor does so with 0.84 precision. In terms of their *break even point (BEP)* measures – the points at which precision equals recall – $GKS_{1d}$ outperforms the other 5 reference tests with a *BEP* of 0.89 while its second best competitor does so with *BEP* of 0.84. Mean precisions for each dataset were compared using the Wilcoxon statistical test with $\alpha = 0.05$. Here, too, $GKS_{1d}$ performs significantly better than all others for both *OCSVM* and *OCNM* versions, except for the *MMD* with a p-value of 0.08 for $GKS_{1d}(OCSVM)$ and 0.12 for $GKS_{1d}(OCNM)$.

Although the plots for our $GKS_{1d}$ (*OCSVM*) test (Figure 4) look better than $GKS_{2d}$, no significant difference was found. This result is consistent with previous studies which claim that variants of solutions whose goal is to make the tests more symmetric have empirically shown no conclusive superiority [4]. We also found that the $GKS_{1d}$ (*OCSVM*) version of our test has the least runtime and scales well with dimensionality, while the $GKS_{1d}$ (*OSNM*) version suffers from increased time complexity, especially in high dimensions, due to its expensive neighborhood measure. However, note that this observation is true only when off-line computational processing on $\mathcal{X}$ is not considered.

As opposed to the KCD, and, PBM, tests, our $GKS_{1d}$ test need not be retrained on each $\mathcal{X}'$. Hence, in the context where $\mathcal{X}$ is treated as a baseline dataset, $GKS_{1d}$ (*OCSVM*) is relatively cheap, and estimated in $O(nm)$ time (the total number of SVs used to calculate $f'_{C(\alpha_1)}, \ldots, f'_{C(\alpha_q)}$ is $O(n)$). In comparison to other tests, it is still the least computationally demanding [7].

## 4.2 Topic Change Detection among Documents

We evaluated our test on an additional setup of high-dimensionality problems pertaining to the detection of topic changes in streams of documents. We used the 20-Newsgroup document corpus [8]. 1000 words were randomly picked to generate 1000 bag-of-words features. 12 categories were used for the experiments [9]. Topic changes were simulated between all pairs of categories (66 pairs in total), using the same methodology as in the previous UCI experiments. Due to the excessive runtime

of some of the tests, especially with high-dimensional data, we evaluated only 4 of the 7 methods: $GKS_{1d}$ (*OCSVM*), *WW*, *MMD*, and *KDQ*, whose expected runtime may be more reasonable.

Once again, our $GKS_{1d}$ test dominates the others with the best precision-recall compromise. With regard to *BEP* values, $GKS_{1d}$ outperforms the other reference tests with a *BEP* of 0.67 (0.70 precision on average), while its second best competitor (*MMD*) does so with a *BEP* of 0.62 (0.64 precision on average). According to the Wilcoxon statistical test with $\alpha = 0.05$, $GKS_{1d}$ performs significantly better than the others in terms of their average precision measures.

## 5  Related Work

Our proposed test belongs to a family of nonparametric tests for detecting change in multivariate data that compare distributions without the intermediate density estimation step. Our reference tests were thus taken from this family of studies. The *kdq-tree test (KDQ)* [4] uses a spatial scheme (called *kdq-tree*) to partition the data into small cells. Then, the Kullback-Leibler (KL) distance is used to measure the difference between data counts for the two samples in each cell. A permutation (bootstrapping) test [7] is used to calculate the significant difference (*p*-value). The *metavariable Wald-Wolfowitz test (WW)* [10] measures the differences between two samples according to the minimum spanning tree in the graph of distances between all pairs in both samples. Then, the Wald-Wolfowitz test statistics are computed over the number of components left in the graph after removing edges between examples of different samples. The *kernel change detection (KCD)* [5] measures the distance between two samples according to a "Fisher-like" distance between samples. This distance is based on hypercircle characteristics of the resulting two *OCSVM*s, which were trained separately on each sample. The *maximum mean discrepancy test (MMD)* [12] meausres discrepancy according to a complete matrix of kernel-based dissimilarity measures between all examples, and test statistics are then computed. (5) The *PAC-Bayesian margin test (PBM)* [6] measures the distance between two samples according to the average margins of a linear SVM classifier between the samples, and test statistics are computed.

As discussed in detail before, our test follows the general approach of Polonik but differs in three important ways: (1) While Polonik uses a density estimator for specifying the MV-sets, we introduce a simpler method that finds the MV-sets directly from the data. Our method is thus more practical and accurate in high-dimensional or small-sample-sized settings. (2) Once the MV-sets are defined, Polonik uses their hypothetical quantiles as the expected plots, and hence, runs the KS test in its one-sample version (goodness-of-fit test). We take a more practically accurate approach for finite sample size when approximations of MV-sets are not precise. Instead of using the hypothetical measures, we estimate the expected plots of $\mathcal{X}$ empirically and use the two-sample KS test instead. (3) Unlike Polonik's work, ours was evaluated empirically and its superiority demonstrated over a wide range of nonparametric tests. Moreover, since Polonik's test relies on a density estimation and the ability to extract its level-sets, it is not practically feasible in high-dimensional settings.

Other methods for estimating MV-sets exist in the literature [21, 1, 16, 13, 20, 23, 14]. Unfortunately, for problems beyond two dimensions and non-convex sets, there is often a gap between their theoretical and practical estimates [20]. We chose here *OCSVM* and *OSNM* because they perform well on small, high-dimensional samples.

## 6  Discussion and Summary

This paper makes two contributions. First, it proposes a new method that uses *OCSVM*s or *OCNM*s to represent high-dimensional distributions as a hierarchy of high-density regions. This method is used for statistical tests, but can also be used as a general, black-box, method for efficient and practical representations of high-dimensional distributions. Second, it presents a nonparametric, generalized, KS test that uses our representation method to detect distributional changes in high-dimensional data. Our test was found superior to competing tests in the sense of average precision and *BEP* measures, especially in the context of change-detection tasks.

An interesting and still open question is how we should set the input $\alpha$ quantiles for our method. The problem of determining the number of quantiles – and the gaps between consecutive ones – is related to the problem of histogram design.

## Footnotes

[1]Note that the separability of the data implies that the solution of Equation 2 satisfies $\rho \neq 0$.

[2]Schölkopf et al. [19] proved that the set provided by *OCSVM* converges asymptotically to the correct probability and not to the correct MV-set. Although this property should be sufficient for the correctness of our test, Polonik observed that MV-sets are preferred.

[3]Note that intersection is still needed (Algorithm 1, line 10) to ensure the hierarchical property on $\hat{C}(\alpha_i)$.

[4]Nominal features were transformed into numeric ones using binary encoding; missing values were replaced by their features' average values.

[5]The code can be downloaded at `http://people.kyb.tuebingen.mpg.de/arthur/mmd.htm`.

[6]Preliminary experiments show similar results obtained with $k$ equal to $10, 20, \ldots, 50\%$ of $|\mathcal{X}|$.

[7]MMD and WW complexities are estimated in $O\left((n+m)^2\right)$ time where $n, m$ are the sample sizes. KDQ uses bootstrapping for *p*-value estimations, and hence, is more expensive.

[8]The 20-Newsgroup corpus is at `http://people.csail.mit.edu/jrennie/20Newsgroups/`.

[9]The selection of these categories is based on the train/test split defined in `http://www.cad.zju.edu.cn/home/dengcai/Data/TextData.html`.

# References

[1] S. Ben-David and M. Lindenbaum. Learning distributions by their density levels: A paradigm for learning without a teacher. *Journal of Computer and System Sciences*, 55(1):171–182, 1997.

[2] ZI Botev, JF Grotowski, and DP Kroese. Kernel density estimation via diffusion. *The Annals of Statistics*, 38(5):2916–2957, 2010.

[3] Chih-Chung Chang and Chih-Jen Lin. *LIBSVM: a library for support vector machines*, 2001.

[4] T. Dasu, S. Krishnan, S. Venkatasubramanian, and K. Yi. An information-theoretic approach to detecting changes in multi-dimensional data streams. In *INTERFACE*, 2006.

[5] F. Desobry, M. Davy, and C. Doncarli. An online kernel change detection algorithm. *Signal Processing, Transactions on Information Theory*, 53(8):2961–2974, 2005.

[6] Anton Dries and Ulrich Rückert. Adaptive concept drift detection. *Statistical Analysis and Data Mining*, 2(5-6):311–327, 2009.

[7] B. Efron and R.J. Tibshirani. *An Introduction to the Bootstrap*. Chapman and Hall/CRC, 1994.

[8] J.H.J. Einmahl and D.M. Mason. Generalized quantile processes. *The Annals of Statistics*, pages 1062–1078, 1992.

[9] G. Fasano and A. Franceschini. A multidimensional version of the kolmogorov-smirnov test. *Monthly Notices of the Royal Astronomical Society*, 225:155–170, 1987.

[10] J.H. Friedman and L.C. Rafsky. Multivariate generalizations of the Wald-Wolfowitz and Smirnov two-sample tests. *The Annals of Statistics*, 7(4):697–717, 1979.

[11] J. Gama, P. Medas, G. Castillo, and P. Rodrigues. Learning with drift detection. In *SBIA*, pages 66–112. Springer, 2004.

[12] A. Gretton, K.M. Borgwardt, M. Rasch, B. Scholkopf, and A.J. Smola. A kernel method for the two-sample-problem. *Machine Learning*, 1:1–10, 2008.

[13] X. Huo and J.C. Lu. A network flow approach in finding maximum likelihood estimate of high concentration regions. *Computational Statistics & Data Analysis*, 46(1):33–56, 2004.

[14] D.M. Mason and W. Polonik. Asymptotic normality of plug-in level set estimates. *The Annals of Applied Probability*, 19(3):1108–1142, 2009.

[15] A. Munoz and J.M. Moguerza. Estimation of high-density regions using one-class neighbor machines. In *PAMI*, pages 476–480, 2006.

[16] J. Nunez Garcia, Z. Kutalik, K.H. Cho, and O. Wolkenhauer. Level sets and minimum volume sets of probability density functions. *International Journal of Approximate Reasoning*, 34(1): 25–47, 2003.

[17] JA Peacock. Two-dimensional goodness-of-fit testing in astronomy. *Monthly Notices of the Royal Astronomical Society*, 202:615–627, 1983.

[18] W. Polonik. Concentration and goodness-of-fit in higher dimensions:(asymptotically) distribution-free methods. *The Annals of Statistics*, 27(4):1210–1229, 1999.

[19] Bernhard Schölkopf, John C. Platt, John C. Shawe-Taylor, Alex J. Smola, and Robert C. Williamson. Estimating the support of a high-dimensional distribution. *Neural Computation*, 13(7):1443–1471, 2001.

[20] C.D. Scott and R.D. Nowak. Learning minimum volume sets. *The Journal of Machine Learning Research*, 7:665–704, 2006.

[21] G. Walther. Granulometric smoothing. *The Annals of Statistics*, pages 2273–2299, 1997.

[22] G. Widmer and M. Kubat. Learning in the presence of concept drift and hidden contexts. *Machine Learning*, 23(1):69–101, 1996.

[23] R.M. Willett and R.D. Nowak. Minimax optimal level-set estimation. *Image Processing, IEEE Transactions on*, 16(12):2965–2979, 2007.

[24] John Wright, Yi Ma, Yangyu Tao, Zhouchen Lin, and Heung-Yeung Shum. Classification via minimum incremental coding length. *SIAM J. Imaging Sciences*, 2(2):367–395, 2009.

